# An Integer Projected Fixed Point Method for Graph Matching and MAP Inference

**Marius Leordeanu**
Robotics Institute
Carnegie Mellon University
Pittsburgh, PA 15213
leordeanu@gmail.com

**Martial Hebert**
Robotics Institute
Carnegie Mellon University
Pittsburgh, PA 15213
hebert@ri.cmu.edu

**Rahul Sukthankar**
Intel Labs Pittsburgh
Pittsburgh, PA 15213
rahuls@cs.cmu.edu

## Abstract

Graph matching and MAP inference are essential problems in computer vision and machine learning. We introduce a novel algorithm that can accommodate both problems and solve them efficiently. Recent graph matching algorithms are based on a general quadratic programming formulation, which takes in consideration both unary and second-order terms reflecting the similarities in local appearance as well as in the pairwise geometric relationships between the matched features. This problem is NP-hard, therefore most algorithms find approximate solutions by relaxing the original problem. They find the optimal continuous solution of the modified problem, ignoring during optimization the original discrete constraints. Then the continuous solution is quickly binarized at the end, but very little attention is put into this final discretization step. In this paper we argue that the stage in which a discrete solution is found is crucial for good performance. We propose an efficient algorithm, with climbing and convergence properties, that optimizes in the discrete domain the quadratic score, and it gives excellent results either by itself or by starting from the solution returned by any graph matching algorithm. In practice it outperforms state-or-the art graph matching algorithms and it also significantly improves their performance if used in combination. When applied to MAP inference, the algorithm is a parallel extension of Iterated Conditional Modes (ICM) with climbing and convergence properties that make it a compelling alternative to the sequential ICM. In our experiments on MAP inference our algorithm proved its effectiveness by significantly outperforming [13], ICM and Max-Product Belief Propagation.

## 1 Introduction

Graph matching and MAP inference are essential problems in computer vision and machine learning that are frequently formulated as integer quadratic programs, where obtaining an exact solution is computationally intractable. We present a novel algorithm, Integer Projected Fixed Point (IPFP), that efficiently finds approximate solutions to such problems. In this paper we focus on graph matching, because it is in this area that we have extensively compared our algorithm to state-of-the-art methods. Feature matching using pairwise constraints is gaining a widespread use in computer vision, especially in shape and object matching and recognition. It is a generalization of the classical graph matching problem, formulated as an integer quadratic program [1,3,4,5,7,8,16,17] that takes into consideration both unary and second-order terms reflecting the similarities in local appearance

as well as in the pairwise geometric relationships between the matched features. The problem is NP-hard, and a lot of effort has been spent in finding good approximate solutions by relaxing the integer one-to-one constraints, such that the continuous global optimum of the new problem can be found efficiently. In the end, little computational time is spent in order to binarize the solution, based on the assumption that the continuous optimum is close to the discrete global optimum of the original combinatorial problem. In this paper we show experimentally that this is not the case and that, in fact, carefully searching for a discrete solution is essential for maximizing the quadratic score. Therefore we propose an iterative algorithm that takes as input any continuous or discrete solution, possibly given by some other graph matching method, and quickly improves it by aiming to maximize the original problem with its integer constraints. Each iteration consists of two stages, being loosely related to the Frank-Wolfe method (FW) [14, 15], a classical optimization algorithm from operation research. The first stage maximizes in the discrete domain a linear approximation of the quadratic function around the current solution, which gives a direction along which the second stage maximizes the original quadratic score in the continuous domain. Even though this second stage might find a non-discrete solution, the optimization direction given by the first stage is always towards an integer solution, which is often the same one found in the second stage. The algorithm always improves the quadratic score in the continuous domain finally converging to a maximum. If the quadratic function is convex the solution at every iteration is always discrete and the algorithm converges in a finite number of steps. In the case of non-convex quadratic functions, the method tends to pass through/near discrete solutions and the best discrete solution encountered along the path is returned, which, in practice is either identical or very close to the point of convergence. We have performed extensive experiments with our algorithm with excellent results, the most representative of which being shown in this paper. Our method clearly outperforms four state-of-the-art algorithms, and, when used in combination, the final solution is dramatically improved. Some recent MAP inference algorithms [11,12,13] for Markov Random Fields formulate the problem as an integer quadratic program, for which our algorithm is also well suited, as we later explain and demonstrate in more detail.

**Matching Using Pairwise Constraints**   The graph matching problem, in its most recent and general form, consists of finding the indicator vector $\mathbf{x}^*$ that maximizes a certain quadratic score function:

**Problem 1:**

$$\mathbf{x}^* = \mathrm{argmax}(\mathbf{x^T M x}) \text{ s. t. } \mathbf{Ax} = \mathbf{1}, \ \mathbf{x} \in \{\mathbf{0, 1}\}^\mathbf{n} \tag{1}$$

given the one-to-one constraints $\mathbf{Ax} = \mathbf{1}$, $\mathbf{x} \in \{0, 1\}^n$, which require that $\mathbf{x}$ is an indicator vector such that $\mathbf{x}_{ia} = 1$ if feature $i$ from one image is matched to feature $a$ from the other image and zero otherwise. Usually one-to-one constraints are imposed on $\mathbf{x}$ such that one feature from one image can be matched to at most one other feature from the other image. In MAP inference problems, only many-to-one constraints are usually required, which can be accommodated by the same formulation, by appropriately setting the constraints matrix $\mathbf{A}$. In graph matching, $\mathbf{M}$ is usually a symmetric matrix with positive elements containing the compatibility score functions, such that $\mathbf{M}_{ia;jb}$ measures how similar the pair of features $(i, j)$ from one image is in both local appearance and pair-wise geometry with the pair of their candidate matches $(a, b)$ from the other image. The difficulty of Problem 1 depends on the structure of this matrix $\mathbf{M}$, but in the general case it is NP-hard and no efficient algorithm exists that can guarantee optimality bounds. Previous algorithms modify Problem 1, usually by relaxing the constraints on the solution, in order to be able to find efficiently optimal solutions to the new problem. For example, spectral matching [5] (SM) drops the constraints entirely and assumes that the leading eigenvector of $\mathbf{M}$ is close to the optimal discrete solution. It then finds the discrete solution $\mathbf{x}$ by maximizing the dot-product with the leading eigenvector of $\mathbf{M}$. The assumption is that $\mathbf{M}$ is a slightly perturbed version of an ideal matrix, with rank-1, for which maximizing this dot product gives the global optimum. Later, spectral graph matching with affine constraints was developed [3] (SMAC), which finds the optimal solution of a modified score function, with a tighter relaxation that imposes the affine constraints $\mathbf{Ax} = \mathbf{1}$ during optimization. A different, probabilistic interpretation, not based on the quadratic formulation, is given in [2] (PM), also based on the assumption that $\mathbf{M}$ is close to a rank-1 matrix, which is the outer product of the vector of probabilities for each candidate assignment. An important observation is that none of the previous methods are concerned with the original integer constraints during optimization, and the final post processing step, when the continuous solution is binarized, is usually just a very simple procedure. They assume that the continuous solution is close to the discrete one. The algorithm

we propose here optimizes the original quadratic score in the continuous domain obtained by only dropping the binary constraints, but it always targets discrete solutions through which it passes most of the time. Note that even in this continuous domain the quadratic optimization problem is NP-hard, so we cannot hope to get any global optimality guarantees. But we do not lose much, since guaranteed global optimality for a relaxed problem does not require closeness to the global optimum of the original problem, a fact that is evident in most of our experiments. Our experimental results from Section 4 strongly suggest an important point: algorithms with global optimality properties in a loosely relaxed domain can often give relatively poor results in the original domain, and a well-designed procedure with local optimality properties in the original domain, such as IPFP, can have a greater impact on the final solution than the global optimality in the relaxed domain.

Our algorithm aims to optimize the following continuous problem, in which we only drop the integer constraints from Problem 1:

**Problem 2:**

$$\mathbf{x}^* = \mathrm{argmax}(\mathbf{x}^\mathbf{T}\mathbf{Mx}) \text{ s. t. } \mathbf{Ax} = \mathbf{1}, \ \mathbf{x} \geq \mathbf{0} \tag{2}$$

Note that Problem 2 is also NP-hard, and it becomes a concave minimization problem, equivalent to Problem 1, when $\mathbf{M}$ is positive definite.

## 2 Algorithm

We introduce our novel algorithm, Integer Projected Fixed Point (IPFP), that takes as input any initial solution, continuous or discrete, and quickly finds a solution obeying the initial discrete constraints of Problem 1 with a better score, most often significantly better than the initial one ($P_d$ from Step 2 is a projection on the discrete domain, discussed shortly afterwards):

1. Initialize $\mathbf{x}^* = \mathbf{x_0}$, $S^* = \mathbf{x_0^T}\mathbf{Mx_0}$, $k = 0$, where $x_i \geq 0$ and $\mathbf{x} \neq \mathbf{0}$
2. Let $\mathbf{b_{k+1}} = P_d(\mathbf{Mx_k})$, $C = \mathbf{x_k^T}\mathbf{M}(\mathbf{b_{k+1}} - \mathbf{x_k})$, $D = (\mathbf{b_{k+1}} - \mathbf{x_k})^\mathbf{T}\mathbf{M}(\mathbf{b_{k+1}} - \mathbf{x_k})$
3. If $D \geq 0$ set $\mathbf{x_{k+1}} = \mathbf{b_{k+1}}$. Else let $r = \min\{-C/D, 1\}$ and set $\mathbf{x_{k+1}} = \mathbf{x_k} + \mathbf{r}(\mathbf{b_{k+1}} - \mathbf{x_k})$
4. If $\mathbf{b_{k+1}^T}\mathbf{Mb_{k+1}} \geq S^*$ then set $S^* = \mathbf{b_{k+1}^T}\mathbf{Mb_{k+1}}$ and $\mathbf{x}^* = \mathbf{b_{k+1}}$
5. If $\mathbf{x_{k+1}} = \mathbf{x_k}$ stop and return the solution $\mathbf{x}^*$
6. Set $k = k + 1$ and go back to Step 2.

This algorithm is loosely related to the power method for eigenvectors, also used by spectral matching [9]: at Step 2 it replaces the fixed point iteration of the power method $\mathbf{v_{k+1}} = P(\mathbf{Mv_k})$, where $P$ is the projection on the unit sphere, with a similar iteration $\mathbf{b_{k+1}} = P_d(\mathbf{Mx_k})$, in which $P_d$ is the projection on the one-to-one (for graph matching) or many-to-one (for MAP inference) discrete constraints. $P_d$ boils down to finding the discrete vector $\mathbf{b_{k+1}} = \mathrm{argmax}\ \mathbf{b^T}\mathbf{Mx_k}$, which can be easily found in linear time for many-to-one constraints. For one-to-one constraints the efficient Hungarian method can be used. This is true since all binary vectors in the given discrete domain have the same norm. Note that (see Proposition 1), in both cases (one-to-one or many-to-one constraints), the discrete $\mathbf{b_{k+1}}$ is also the one maximizing the dot-product with $\mathbf{Mx_k}$ in the continuous domain $\mathbf{Ab} = \mathbf{1}, \mathbf{b} > \mathbf{0}$. IPFP is also related to Iterative Conditional Modes (ICM) [10] used for inference in graphical models. In the domain of many-to-one constraints IPFP becomes an extension of ICM for which the updates are performed in parallel without losing the climbing property and the convergence to a discrete solution. Note that the fully parallel version of ICM is IPFP without Step 3: $\mathbf{x_{k+1}} = P_d(\mathbf{Mx_k})$. The theoretical results that we will present shortly are valid for both one-to-one and many-to-one constraints, with a few differences that we will point out when deemed necessary.

The algorithm is a basically a sequence of linear assignment (or independent labeling) problems, in which the next solution is found by using the previous one. In practice the algorithm converges in about $5 - 10$ steps, which makes it very efficient, with basically the same complexity as the complexity of Step 2. Step 3 insures that the quadratic score increases with each iteration. Step 4 guarantees that the binary solution returned is never worse than the initial solution. In practice, the algorithm significantly improves the initial binary solution, and the final continuous solution is most often discrete, and always close to the best discrete one found. In fact, in the case of MAP inference, it is guaranteed that the point of convergence is discrete, as a fixed point of $P_d$.

**Intuition**  The intuition behind this algorithm is the following: at every iteration the quadratic score $\mathbf{x^T M x}$ is first approximated by the first order Taylor expansion around the current solution $\mathbf{x_k}$: $\mathbf{x^T M x} \approx \mathbf{x_k^T M x_k} + 2\mathbf{x_k^T M}(\mathbf{x} - \mathbf{x_k})$. This approximation is maximized within the discrete domain of Problem 1, at Step 2, where $\mathbf{b_{k+1}}$ is found. From Proposition 1 (see next) we know that the same discrete $\mathbf{b_{k+1}}$ also maximizes the linear approximation in the continuous domain of Problem 2. The role of $\mathbf{b_{k+1}}$ is to provide a direction of largest possible increase (or ascent) in the first-order approximation, within both the continuous domain and the discrete domain simultaneously. Along this direction the original quadratic score can be further maximized in the continuous domain of Problem 2 (as long as $\mathbf{b_{k+1}} \neq \mathbf{x_k}$). At Step 3 we find the optimal point along this direction, also inside the continuous domain of Problem 2. The hope, also confirmed in practice, is that the algorithm will tend to converge towards discrete solutions that are, or are close to, maxima of Problem 2.

# 3  Theoretical Analysis

**Proposition 1:** *For any vector* $\mathbf{x} \in R^n$ *there exists a global optimum* $\mathbf{y}^*$ *of* $\mathbf{x^T M y}$ *in the domain of Problem* 2 *that has binary elements (thus it is also in the domain of Problem* 1*).*

**Proof:** Maximizing $\mathbf{x^T M y}$ with respect to $\mathbf{y}$, subject to $\mathbf{Ay} = \mathbf{1}$ and $\mathbf{y} > 0$ is a linear program for which an integer optimal solution exists because the constraints matrix $\mathbf{A}$ is totally unimodular [9]. This is true for both one-to-one and many-to-one constraints.

It follows that the maximization from Step 2 $\mathbf{b_{k+1}} = \operatorname{argmax} \mathbf{b^T M x_k}$ in the original discrete domain, also maximizes the same dot-product in the continuous domain of Problem 2, of relaxed constraints $\mathbf{Ax} = \mathbf{1}$ and $\mathbf{x} > 0$. This ensures that the algorithm will always move towards some discrete solution that also maximizes the linear approximation of the quadratic function in the domain of Problem 2. Most often in practice, that discrete solution also maximizes the quadratic score, along the same direction and within the continuous domain. Therefore $\mathbf{x_k}$ is likely to be discrete at every step.

**Property 1:**

*The quadratic score* $\mathbf{x_k^T M x_k}$ *increases at every step* $k$ *and the sequence of* $\mathbf{x_k}$ *converges.*

**Proof:**

For a given step $k$, if $\mathbf{b_{k+1}} = \mathbf{x_k}$ we have convergence. If $\mathbf{b_{k+1}} \neq \mathbf{x_k}$, let $\mathbf{x}$ be a point on the line between $\mathbf{x_k}$ and $\mathbf{b_{k+1}}$, $\mathbf{x} = \mathbf{x_k} + t(\mathbf{b_{k+1}} - \mathbf{x_k})$. For any $0 \leq t \leq 1$, $\mathbf{x}$ is in the feasible domain of Problem 2. Let $S_k = \mathbf{x_k^T M x_k}$. Let us define the quadratic function $f(t) = \mathbf{x^T M x} = S_k + 2tC + t^2 D$, which is the original function in the domain of Problem 2 on the line between $\mathbf{x_k}$ and $\mathbf{b_{k+1}}$. Since $\mathbf{b_{k+1}}$ maximizes the dot product with $\mathbf{x_k^T M}$ in the discrete (and the continuous) domain, it follows that $C \geq 0$. We have two cases: $D \geq 0$, when $\mathbf{x_{k+1}} = \mathbf{b_{k+1}}$ (Step 3) and $S_{k+1} = \mathbf{x_{k+1}^T M x_{k+1}} = f_q(1) \geq S_k = \mathbf{x_k^T M x_k}$; and $D < 0$, when the quadratic function $f_q(t)$ is convex with the maximum in the domain of Problem 2 attained at point $\mathbf{x_{k+1}} = \mathbf{x_k} + \mathbf{r}(\mathbf{b_{k+1}} - \mathbf{x_k})$. Again, it also follows that $S_{k+1} = \mathbf{x_{k+1}^T M x_{k+1}} = f_q(r) \geq S_k = \mathbf{x_k^T M x_k}$. Therefore, the algorithm is guaranteed to increase the score at every step. Since the score function is bounded above on the feasible domain, it has to converge, which happens when $C = 0$.

By always improving the quadratic score in the continuous domain, at each step the next solution moves towards discrete solutions that are better suited for solving the original Problem 1.

**Property 2:** *The algorithm converges to a maximum of Problem* 2.

**Proof:**

Let $\mathbf{x}^*$ be a point of convergence. At that point the gradient $2\mathbf{M x}^*$ is non-zero since both $\mathbf{M}$ and $\mathbf{x}^*$ have positive elements and $(\mathbf{x}^*)^T \mathbf{M x}^* > 0$, (it is higher than the score at the first iteration, also greater than zero). Since $\mathbf{x}^*$ is a point of convergence it follows that $C = 0$, that is, for any other $\mathbf{x}$ in the continuous domain of Problem 2, $(\mathbf{x}^*)^T \mathbf{M x}^* \geq (\mathbf{x}^*)^T \mathbf{M x}$. This implies that for any direction vector $\mathbf{v}$ such that $\mathbf{x}^* + t\mathbf{v}$ is in the domain of Problem 2 for a small enough $t > 0$, the dot-product between $\mathbf{v}$ and the gradient of the quadratic score is less than or equal to zero $(\mathbf{x}^*)^T \mathbf{M v} \leq 0$, which further implies that $\mathbf{x}^*$ is a maximum (local or global) of the quadratic score within the continuous domain of equality constraints $\mathbf{Ax}^* = \mathbf{1}$, $\mathbf{x}^* > \mathbf{0}$.

For many-to-one constraints (MAP inference) it basically follows that the algorithm will converge to a discrete solution, since the strict (local and global) maxima of Problem 2 are in the discrete domain [12]. If the maximum is not strict, IPFP still converges to a discrete solution (which is also a local maximum): the one found at Step 2. This is another similarity with ICM, which also converges to a maximum. Therefore, combining ours with ICM cannot improve the performance of ICM, and vice-versa.

**Property 3:** *If $\mathbf{M}$ is positive semidefinite with positive elements, then the algorithm converges in a finite number of iterations to a discrete solution, which is a maximum of Problem 2.*

**Proof:** Since $\mathbf{M}$ is positive semidefinite we always have $D \geq 0$, thus $\mathbf{x_k}$ is always discrete for any $k$. Since the number of discrete solutions is finite, the algorithm must converge in a finite number of steps to a local (or global) maximum, which must be discrete. This result is obviously true for both one-to-one and many-to-one constraints.

When $\mathbf{M}$ is positive semidefinite, Problem 2 is a concave minimization problem for which it is well known that the global optimum has integer elements, so it is also a global optimum of the original Problem 1. In this case our algorithm is only guaranteed to find a local optimum in a finite number of iterations. Global optimality of concave minimization problems is a notoriously difficult task since the problem can have an exponential number of local optima. In fact, if a large enough constant is added to the diagonal elements of $\mathbf{M}$, every point in the original domain of possible solutions becomes a local optimum for one-to-one problems. Therefore adding a large constant to make the problem concave is not good idea , even if the global optimum does not change. In practice $\mathbf{M}$ is rarely positive semidefinite, but it can be close to being one if the first eigenvalue is much larger than the rest, which is the assumption made by the spectral matching algorithm, for example.

**Property 4:** *If $\mathbf{M}$ has non-negative elements and is rank-1, then the algorithm will converge and return the global optimum of the original problem after the first iteration.*

**Proof:**

Let $\mathbf{v}, \lambda$ be the leading eigenpair of $\mathbf{M}$. Then, since $\mathbf{M}$ has non-negative elements both $\mathbf{v}$ and $\lambda$ are positive. Since $\mathbf{M}$ is also rank one, we have $\mathbf{M}\mathbf{x}_0 = \lambda(\mathbf{v}^T\mathbf{x}_0)\mathbf{v}$. Since both $\mathbf{x}_0$ and $\mathbf{v}$ have positive elements it immediately follows that $\mathbf{x}_1$ after the first iteration is the indicator solution vector that maximizes the dot-product with the leading eigenvector ($\mathbf{v}^T\mathbf{x}_0 = 0$ is a very unlikely case that never happens in practice). It is clear that this vector is the global optimum, since in the rank-1 case we have: $\mathbf{x}^T\mathbf{M}\mathbf{x} = \lambda_1(\mathbf{v}^T\mathbf{x})^2$, for any $\mathbf{x}$.

The assumption that $\mathbf{M}$ is close to being rank-1 is used by two recent algorithms, [2] and [5]. Spectral matching [5] also returns the optimal solution in this case and it assumes that the rank-1 assumption is the ideal matrix to which a small amount of noise is added. Probabilistic graph matching [2] makes the rank-1 approximation by assuming that each second-order element of $\mathbf{M}_{ia;jb}$ is the product of the probability of feature $i$ being matched to $a$ and feature $j$ being matched to $b$, independently. However, instead of maximizing the quadratic score function, they use this probabilistic interpretation of the pair-wise terms and find the solution by looking for the closest rank-1 matrix to $\mathbf{M}$ in terms of the KL-divergence. If the assumptions in [2] were perfectly met, then spectral matching, probabilistic graph matching and our algorithm would all return the same solution. For a comparison of all these algorithms on real world experiments please see the experiments section.

## 4  Experiments

We first present some representative experiments on graph matching problems. We tested IPFP by itself, as well as in conjunction with other algorithms as a post-processing step. When used by itself IPFP is always initialized with a flat, uniform continuous solution. We followed the experiments of [6] in the case of outliers: we used the same cars and motorbikes image pairs, extracted from the Pascal 2007 database, the same features (oriented points extracted from contours) and the same second-order potentials $\mathbf{M}_{ia;jb} = e^{-\mathbf{w}^T \mathbf{g}_{ia;jb}}$; $\mathbf{g}_{ia;jb}$ is a five dimensional vector of deformations in pairwise distances and angles when matching features $(i, j)$ from one image to features $(a, b)$ from the other and $\mathbf{w}$ is the set of parameters that control the weighting of the elements of $\mathbf{g}_{ia;jb}$. We followed the setup from [6] exactly, in order to have a fair comparison of our algorithm against the results they obtained. Due to space limitations, we refer the interested reader to [6] for the details. These experiments are difficult due the large number of outliers (on average 5 times more outliers than inliers), and, in the case of cars and motorbikes, also due to the large intra-category variations

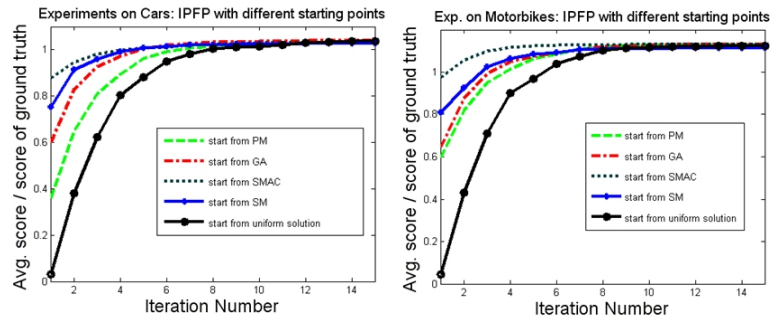

Figure 1: Results on motorbikes and cars averaged over 30 experiments: at each iteration the average score $\mathbf{x_k^T M x_k}$ normalized by the ground truth score is displayed. The comparisons are not affected by this normalization, since all scores are normalized by the same value. Notice how quickly IPFP converges (fewer than 10 iterations)

Table 1: Average matching rates for the experiments with outliers on cars and motorbikes from Pascal 07. Note that our algorithm by itself outperforms on average all the others by themselves. When the solution of other algorithms is the starting point of IPFP the performance is greatly improved.

| Dataset | IPFP | SM | SMAC | GA | PM |
|---|---|---|---|---|---|
| Cars and Motorbikes: alone | **64.4%** | 58.2% | 58.6% | 46.7% | 36.6% |
| Cars and Motorbikes: + IPFP | 64.4% | 67.0% | 66.2% | 66.3% | **67.2%** |
| Cars and Motorbikes: Improvement | NA | +8.8% | +7.6% | +19.6% | +30.6% |

in shape present in the Pascal 2007 database. By outliers we mean the features that have no ground truth correspondences in the other image, and by inliers those that have such correspondences. As in [6] we allow outliers only in one of the images in which they are present in large number, the ratio of outliers to inliers varying from 1.5 to over 10. The ground truth correspondences were manually selected by the authors of [6].

The difficulty of the matching problems is reflected by the relatively low matching scores of all algorithms (Table 1). In order to ensure an optimal performance of all algorithms, we used the supervised version of the graph matching learning method from [6]. Learning $\mathbf{w}$ was effective, improving the performance by more than $15\%$ on average, for all algorithms. The algorithms we chose for comparison and also for combining with ours are among the current state-of-the-art in the literature: spectral matching with affine constraints (SMAC) [3], spectral matching (SM) [5], probabilistic graph matching (PM) [2], and graduated assignment (GA) [4]. In Tables 1 and 2 we show that in our experiments IPFP significantly outperforms other state-of-the-art algorithms.

In our experiments we focused on two aspects. Firstly, we tested the matching rate of our algorithm against the others, and observed that it consistently outperforms them, both in the matching rate and in the final quadratic score achieved by the resulting discrete solution (see Tables 1, 2). Secondly, we combined our algorithm, as a post-processing step, with the others and obtained a significant improvement over the output matching rate and quadratic score of the other algorithms by themselves (see Figures 1, 2). In Figure 2 we show the quadratic score of our algorithm, per iteration, for several individual experiments, when it takes as initial solution the output of several other algorithms. The score at the first iteration is the score of the final discrete solution returned by those algorithms and the improvement in just a few iterations is substantial, sometimes more than doubling the final quadratic score reached by the other algorithms. In Figure 1 we show the average scores of our algorithm, over 30 different experiments on cars and motorbikes, per iteration, normalized by the score of the solutions given by the human ground truth labeling. We notice that regardless of the starting condition, the final scores are very similar, slightly above the value of 1 (Table 2), which means that the solutions reached are, on average, at least as good, in terms of the matching score function, as the manually picked solutions. None of the algorithms by themselves, except only for IPFP, reach this level of quality. We also notice that a quadratic score of 1 does not correspond to a perfect matching rate, which indicates the fact that besides the ground truth solution there are

Table 2: Quadratic scores on the Cars and Motorbikes image sets (the higher, the better). $S^*$ is the score of the manually picked ground truth. Note that the ground truth score $S^*$ does not affect the comparison since it is the same normalization value for all algorithms. The "Convergence to a binary solution" row shows the average rate at which our algorithm converges to a discrete solution.

| Experiments on Cars and Motorbikes | IPFP | SM | SMAC | GA | PM |
|---|---|---|---|---|---|
| Alone, avg $S_{max}/S^*$ | **1.081** | 0.781 | 0.927 | 0.623 | 0.4785 |
| + IPFP, avg $S_{max}/S^*$ | 1.081 | 1.070 | 1.082 | **1.086** | 1.080 |
| Convergence to a binary solution | 86.7% | 93.3% | 86.7% | 93.3% | 86.7% |

other solutions with high score. This is expected, given that the large number of outliers can easily introduce wrong solutions of high score. However, increasing the quadratic score, does increase the matching rate as can be seen by comparing the results between the Tables 2 and 1.

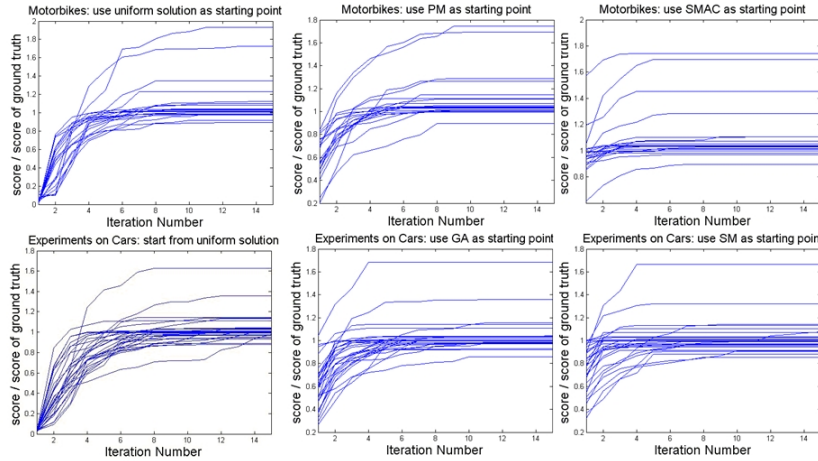

Figure 2: Experiments on cars and motorbikes: at each iteration the score $\mathbf{x_k^T M x_k}$ normalized by the ground truth score is displayed for $30$ individual matching experiments for our algorithm starting from different solutions (uniform, or given by some other algorithm).

**Experiments on MAP inference problems**   We believe that IPFP can have a greater impact in graph matching problems than in MAP inference ones, due to the lack of efficient, high-quality discretization procedures in the graph matching literature. In the domain of MAP inference for MRFs, it is important to note that IPFP is strongly related to the parallel version of Iterated Conditional Modes, but, unlike parallel ICM, it has climbing, strong convergence and local optimality properties. To see the applicability of our method to MAP inference, we tested it against sequential ICM, Max-Product BP with damping oscillations (Table 3), the algorithm L2QP of [12], and the the algorithm of [13], which is based on a convex approximation. In the case of [12] and [13], which give continuous optimal solutions to a relaxed problem, a post-processing step is required for discretization. Note that the authors of [13] use ICM to obtain a binary solution. However, we wanted to emphasize the quality of the methods by themselves, without a powerful discretization step, and used ICM for comparisons separately. Thus, for discretization we used one iteration of ICM for both our L2QP [12] and CQP [13]. Both ICM and IPFP used as initial condition a uniform flat solution as in the case of graph matching. We used the same experimental setup as in [11] and [12], on graphs with different degrees of edge density (by generating random edges with a given probability, varying from $0.1$ to $1$). The values of the potentials were randomly generated as in [11] and [12], favoring the correct labels vs. the wrong ones. In Figure 3 we show the average scores normalized by the score of IPFP over $30$ different experiments, for different probabilities of edge generation $pEdge$ on graphs with $50$ nodes and different number of possible labels per node. The most important observation is that both ICM and IPFP outperform L2QP and CQP by a wide margin on all problems without any single exception. In our experiments, on every single problem, IPFP

Table 3: Average objective score over 30 different experiments on 4-connected and 8-connected planar graphs with 50 sites and 10 possible labels per site

| Graph type | IPFP | ICM | BP |
|---|---|---|---|
| 4-connected planar | 79.5 | 78.2 | 54.2 |
| 8-connected planar | 126.0 | 123.2 | 75.4 |

outperformed ICM, while both IPFP and ICM outperformed both L2QP and CQP by a wide margin, which is reflected in the averages shown in Figure 3.

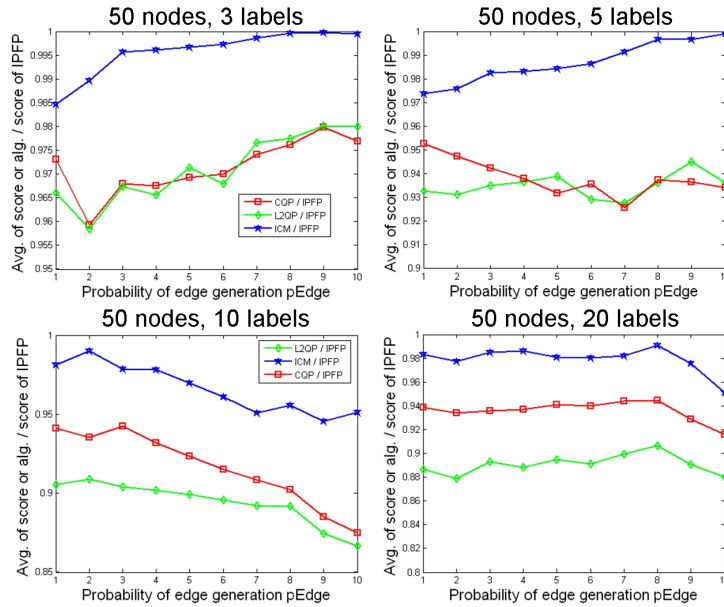

Figure 3: Average quadratic scores normalized by the score of IPFP, over 30 different experiments, for each probability of edge generation $pEdge \in 0.1, 0.2, ..., 1$ and different number of labels, for graphs with 50 nodes. Note that IPFP consistently ourperforms L2QP [12] and CQP [13] (by a wide margin) and ICM. Note that L2QP and CQP perform similarly for a small number of labels.

## 5   Conclusion

This paper presents a novel and computationally efficient algorithm, Integer Projected Fixed Point (IPFP), that outperforms state-of-the-art methods for solving quadratic assignment problems in graph matching, and well-established methods in MAP inference such as BP and ICM. We analyze the theoretical properties of IPFP and show that it has strong convergence and climbing guarantees. Also, IPFP can be employed in conjunction with existing techniques, such as SMAC or SM for graph matching or BP for inference to achieve solutions that are dramatically better than the ones produced independently by those methods alone. Furthermore, IPFP is very straightforward to implement and converges in only 5–10 iterations in practice. Thus, IPFP is very well suited for addressing a broad range of real-world problems in computer vision and machine learning.

## 6   Acknowledgments

This work was supported in part by NSF Grant IIS0713406 and by the Intel Graduate Fellowship program.

# References

[1] A. Berg, T. Berg and J. Malik. Shape matching and object recognition using low distortion correspondences. *Computer Vision and Pattern Recognition*, 2005

[2] R. Zass and A. Shashua. Probabilistic Graph and Hypergraph Matching. *Computer Vision and Pattern Recognition*, 2008

[3] T. Cour, P. Srinivasan and J. Shi. Balanced Graph Matching. *Neural Information Processing Systems*, 2006

[4] S. Gold, and A. Rangarajan. A graduated assignment algorithm for graph matching. *Pattern Analysis and Machine Intelligence*, 1996

[5] M. Leordeanu and M. Hebert. A Spectral Technique for Correspondence Problems using Pairwise Constraints. *International Conference on Computer Vision*, 2005

[6] M. Leordeanu and M. Hebert. Unsupervised Learning for Graph Matching. *Computer Vision and Pattern Recognition*, 2009

[7] C. Schellewald and C. Schnorr. Probabilistic subgraph matching based on convex relaxation. *EMMCVPR*, 2005

[8] P.H.S Torr. Solving markov random fields using semi definite programming. *Artificial Intelligence and Statistics*, 2003

[9] B. Rainer, M. Dell'Amico and S. Martello. Assignment Problems. *SIAM Publications*, 2009

[10] J. Besag. On the Statistical Analysis of Dirty Pictures. *JRSS*, 1986

[11] T. Cour and J. Shi. Solving Markov Random Fields with Spectral Relaxation. *International Conference on Artificial Intelligence and Statistics*, 2007

[12] M. Leordeanu and M. Hebert. Efficient MAP approximation for dense energy functions. *International Conference on Machine Learning*, 2006

[13] P. Ravikumar and J. Lafferty. Quadratic Programming Relaxations for Metric Labeling and Markov Random Field MAP Estimation, *International Conference on Machine Learning*, 2006

[14] M. Frank and P. Wolfe. An algorithm for quadratic programming, *Naval Research Logistics Quarterly*, 1956.

[15] N.W. Brixius and K.M. Anstreicher. Solving quadratic assignment problems using convex quadratic programming relaxations, *Optimization Methods and Software*, 2001

[16] J. Maciel and J.P. Costeira. A global solution to sparse correspondence problems *Pattern Analysis and Machine Intelligence*, 2003

[17] L. Torresani, V. Kolmogorov and C. Rother. Feature correspondence via graph matching: Models and global optimization. *European Conference on Computer Vision*, 2008
